# Multi-Task Learning via Conic Programming

**Tsuyoshi Kato**[*,°]**, Hisashi Kashima**[†]**, Masashi Sugiyama**[‡]**, Kiyoshi Asai**[*,°]
[*] Graduate School of Frontier Sciences, The University of Tokyo,
[°] Institute for Bioinformatics Research and Development (BIRD),
Japan Science and Technology Agency (JST)
[†] Tokyo Research Laboratory, IBM Research,
[‡] Department of Computer Science, Tokyo Institute of Technology,
[°] AIST Computational Biology Research Center,
kato-tsuyoshi@cb.k.u-tokyo.ac.jp,  kashi_pong@yahoo.co.jp,
sugi@cs.titech.ac.jp,  asai@cbrc.jp

## Abstract

When we have several related tasks, solving them simultaneously is shown to be
more effective than solving them individually. This approach is called *multi-task
learning* (MTL) and has been studied extensively. Existing approaches to MTL
often treat all the tasks as *uniformly* related to each other and the relatedness of
the tasks is controlled *globally*. For this reason, the existing methods can lead
to undesired solutions when some tasks are not highly related to each other, and
some pairs of related tasks can have significantly different solutions. In this pa-
per, we propose a novel MTL algorithm that can overcome these problems. Our
method makes use of a *task network*, which describes the relation structure among
tasks. This allows us to deal with intricate relation structures in a systematic way.
Furthermore, we control the relatedness of the tasks *locally*, so all pairs of related
tasks are guaranteed to have similar solutions. We apply the above idea to sup-
port vector machines (SVMs) and show that the optimization problem can be cast
as a *second order cone program*, which is convex and can be solved efficiently.
The usefulness of our approach is demonstrated through simulations with protein
super-family classification and ordinal regression problems.

## 1   Introduction

In many practical situations, a classification task can often be divided into *related* sub-tasks. Since
the related sub-tasks tend to share common factors, solving them together is expected to be more
advantageous than solving them independently. This approach is called *multi-task learning* (MTL,
a.k.a. *inductive transfer* or *learning to learn*) and has theoretically and experimentally proven to be
useful [4, 5, 8].

Typically, the 'relatedness' among tasks is implemented as imposing the solutions of related tasks to
be similar (e.g. [5]). However, the MTL methods developed so far have several limitations. First, it
is often assumed that all sub-tasks are related to each other [5]. However, this may not be always true
in practice—some are related but others may not be. The second problem is that the related tasks
are often imposed to be close in the sense that the *sum* of the distances between solutions over all
pairs of related tasks is upper-bounded [8] (which is often referred to as the *global* constraint [10]).
This implies that all the solutions of related tasks are not necessarily close, but some can be quite
different.

In this paper, we propose a new MTL method which overcomes the above limitations. We settle the
first issue by making use of a *task network* that describes the relation structure among tasks. This
enables us to deal with intricate relation structures in a systematic way. We solve the second problem

by directly upper-bounding *each* distance between the solutions of related task pairs (which we call *local* constraints).

We apply this ideas in the framework of support vector machines (SVMs) and show that linear SVMs can be trained via a *second order cone program* (SOCP) [3] in the primal. An SOCP is a convex problem and the global solution can be computed efficiently. We further show that the kernelized version of the proposed method can be formulated as a *matrix-fractional program* (MFP) [3] in the dual, which can be again cast as an SOCP; thus the optimization problem of the kernelized variant is still convex and the global solution can be computed efficiently. Through experiments with artificial and real-world protein super-family classification data sets, we show that the proposed MTL method compares favorably with existing MTL methods.

We further test the performance of the proposed approach in *ordinal regression* scenarios [9], where the goal is to predict ordinal class labels such as users' preferences ('like'/'neutral'/'dislike') or students' grades (from 'A' to 'F'). The ordinal regression problems can be formulated as a set of one-versus-one classification problems, e.g., 'like' vs. 'neutral' and 'neutral' vs. 'dislike'. In ordinal regression, the relatedness among tasks is highly structured. That is, the solutions (decision boundaries) of *adjacent* problems are expected to be similar, but others may not be related, e.g., 'A' vs. 'B' and 'B' vs. 'C' would be related, but 'A' vs. 'B' and 'E' vs. 'F' may not be. Our experiments demonstrate that the proposed method is also useful in the ordinal regression scenarios and tends to outperform existing approaches [9, 8]

## 2    Problem Setting

In this section, we formulate the MTL problem.

Let us consider $M$ binary classification tasks, which all share the common input-output space $\mathcal{X} \times \{\pm 1\}$. For the time being, we assume $\mathcal{X} \subset \mathbb{R}^d$ for simplicity; later in Section 4, we extend it to reproducing kernel Hilbert spaces. Let $\{\boldsymbol{x}_t, y_t\}_{t=1}^{\ell}$ be the training set, where $\boldsymbol{x}_t \in \mathcal{X}$ and $y_t \in \{\pm 1\}$ for $t = 1, \ldots, \ell$. Each data sample $(\boldsymbol{x}_t, y_t)$ has its target task; we denote the set of sample indices of the $i$-th task by $\mathcal{I}_i$. We assume that each sample belongs only to a single task, i.e., the index sets are exclusive: $\sum_{i=1}^{M} |\mathcal{I}_i| = \ell$ and $\mathcal{I}_i \cap \mathcal{I}_j = $ null, $\forall i \neq j$.

The goal is to learn the score function of each classification task: $f_i(\boldsymbol{x}; \boldsymbol{w}_i, b_i) = \boldsymbol{w}_i^\top \boldsymbol{x} + b_i$, for $i = 1, \ldots, M$, where $\boldsymbol{w}_i \in \mathbb{R}^d$ and $b_i \in \mathbb{R}$ are the model parameters of the $i$-th task. We assume that a *task network* is available. The task network describes the relationships among tasks, where each node represents a task and two nodes are connected by an edge if they are related to each other [1]. We denote the edge set by $\mathcal{E} \equiv \{(i_k, j_k)\}_{k=1}^{K}$.

## 3    Local MTL with Task Network: Linear Version

In this section, we propose a new MTL method.

### 3.1    Basic Idea

When the relation among tasks is not available, we may just solve $M$ penalized fitting problems individually:

$$\frac{1}{2}\|\boldsymbol{w}_i\|^2 + C_\alpha \sum_{t \in \mathcal{I}_i} \text{Hinge}(f_i(\boldsymbol{x}_t; \boldsymbol{w}_i, b_i), y_t), \qquad \text{for } i = 1, \ldots, M, \qquad (1)$$

where $C_\alpha \in \mathbb{R}_+$ is a regularization constant and $\text{Hinge}(\cdot, \cdot)$ is the *hinge* loss function: $\text{Hinge}(f, y) \equiv \max(1 - fy, 0)$. This individual approach tends to perform poorly if the number of training samples in each task is limited—the performance is expected to be improved if more training samples are available. Here, we can exploit the information of the task network. A naive

idea would be to use the training samples of neighboring tasks in the task network for solving the target fitting problem. However, this does not fully make use of the network structure since there are many other *indirectly* connected tasks via some paths on the network.

To cope with this problem, we take another approach here, which is based on the expectation that the *solutions* of related tasks are close to each other. More specifically, we impose the following constraint on the optimization problem (1):

$$\frac{1}{2} \|\boldsymbol{w}_{i_k} - \boldsymbol{w}_{j_k}\|^2 \leq \rho, \quad \text{for } \forall k = 1, \ldots, K. \tag{2}$$

Namely, we upper-bound each difference between the solutions of related task pairs by a positive scalar $\rho \in \mathbb{R}_+$. We refer to this constraint as *local constraint* following [10]. Note that we do not impose a constraint on the bias parameter $b_i$ since the bias could be significantly different even among related tasks. The constraint (2) allows us to *implicitly* increase the number of training samples over the task network in a systematic way through the solutions of related tasks.

Following the convention [8], we blend Eqs.(1) and (2) as

$$\frac{1}{2M} \sum_{i=1}^{M} \|\boldsymbol{w}_i\|^2 + C_\alpha \sum_{i=1}^{M} \sum_{t \in \mathcal{I}_i} \text{Hinge}(f_i(\boldsymbol{x}_t; \boldsymbol{\theta}), y_t) + C_\rho \rho, \tag{3}$$

where $C_\rho$ is a positive trade-off parameter. Then our optimization problem is summarized as follows:

**Problem 1.**

$$\min \quad \frac{1}{2M} \sum_{i=1}^{M} \|\boldsymbol{w}_i\|^2 + C_\alpha \|\boldsymbol{\xi}\|_1 + C_\rho \rho, \quad \text{wrt.} \quad \boldsymbol{w} \in \mathbb{R}^{Md}, \ \boldsymbol{b} \in \mathbb{R}^M, \ \boldsymbol{\xi}_\alpha \in \mathbb{R}_+^\ell, \ \text{and } \rho \in \mathbb{R}_+,$$

$$\text{subj. to} \quad \frac{1}{2} \|\boldsymbol{w}_{i_k} - \boldsymbol{w}_{j_k}\|^2 \leq \rho, \ \forall k, \qquad \text{and} \quad y_t\left(\boldsymbol{w}_i^\top \boldsymbol{x}_t + b_i\right) \geq 1 - \xi_t^\alpha, \ \forall t \in \mathcal{I}_i, \forall i$$

$$\text{where} \quad \boldsymbol{w} \equiv \left[\boldsymbol{w}_1^\top, \ldots, \boldsymbol{w}_M^\top\right]^\top, \qquad \text{and} \quad \boldsymbol{\xi}_\alpha \equiv [\xi_1^\alpha, \ldots, \xi_\ell^\alpha]^\top. \tag{4}$$

### 3.2 Primal MTL Learning by SOCP

The *second order cone program* (SOCP) is a class of convex programs of minimizing a linear function over an intersection of second-order cones [3]:[2]

**Problem 2.**

$$\min \quad \boldsymbol{f}^\top \boldsymbol{z} \qquad \text{wrt} \quad \boldsymbol{z} \in \mathbb{R}^n \quad \text{subj. to} \quad \|\boldsymbol{A}_i \boldsymbol{z} + \boldsymbol{b}_i\| \leq \boldsymbol{c}_i^\top \boldsymbol{z} + d_i, \quad \text{for } i = 1, \ldots, N, \tag{5}$$

$$\text{where } \boldsymbol{f} \in \mathbb{R}^n, \ \boldsymbol{A}_i \in \mathbb{R}^{(n_i-1) \times n}, \ \boldsymbol{b}_i \in \mathbb{R}^{n_i-1}, \ \boldsymbol{c}_i \in \mathbb{R}^n, \ d_i \in \mathbb{R}.$$

Linear programs, quadratic programs, and quadratically-constrained quadratic programs are actually special cases of SOCPs. SOCPs are a sub-class of semidefinite programs (SDPs) [3], but SOCPs can be solved more efficiently than SDPs. Successful optimization algorithms for both SDP and SOCP are interior-point algorithms. The SDP solvers (e.g. [2]) consume $O(n^2 \sum_i n_i^2)$ time complexity for solving Problem 2, but the SOCP-specialized solvers that directly solve Problem 2 take only $O(n^2 \sum_i n_i)$ computation [7]. Thus, SOCPs can be solved more efficiently than SDPs.

We can show that Problem 1 is cast as an SOCP using hyperbolic constraints [3].

**Theorem 1.** *Problem 1 can be reduced to an SOCP and it can be solved with $O((Md+\ell)^2(Kd+\ell))$ computation.*

## 4 Local MTL with Task Network: Kernelization

The previous section showed that a linear version of the proposed MTL method can be cast as an SOCP. In this section, we show how the kernel trick could be employed for obtaining a non-linear variant.

## 4.1 Dual Formulation

Let $\boldsymbol{K}_{\text{fea}}$ be a positive semidefinite matrix with the $(s,t)$-th element being the inner-product of feature vectors $\boldsymbol{x}_s$ and $\boldsymbol{x}_t$: $K_{s,t}^{\text{fea}} \equiv \langle \boldsymbol{x}_s, \boldsymbol{x}_t \rangle$. This is a kernel matrix of feature vectors. We also introduce a kernel among tasks. Using a new $K$-dimensional non-negative parameter vector $\boldsymbol{\lambda} \in \mathbb{R}_+^K$, we define the kernel matrix of tasks by

$$\boldsymbol{K}_{\text{net}}(\boldsymbol{\lambda}) \equiv \left( \frac{1}{M} \boldsymbol{I}_M + \mathcal{U}\boldsymbol{\lambda} \right)^{-1},$$

where $\mathcal{U}\boldsymbol{\lambda} \equiv \sum_{k=1}^K \lambda_k \boldsymbol{U}_k$, $\boldsymbol{U}_k \equiv \boldsymbol{E}^{i_k i_k} + \boldsymbol{E}^{j_k j_k} - \boldsymbol{E}^{i_k j_k} - \boldsymbol{E}^{j_k i_k}$, and $\boldsymbol{E}^{(i,j)} \in \mathbb{R}^{M \times M}$ is the sparse matrix whose $(i,j)$-th element is one and all the others are zero. Note that this is the graph Laplacian kernel [11], where the $k$-th edge is weighted according to $\lambda_k$. Let $\boldsymbol{Z} \in \mathbb{N}^{M \times \ell}$ be the indicator of a task and a sample such that $Z_{i,t} = 1$ if $t \in \mathcal{I}_i$ and $Z_{i,t} = 0$ otherwise. Then the information about the tasks are expressed by the $\ell \times \ell$ kernel matrix $\boldsymbol{Z}^\top \boldsymbol{K}_{\text{net}}(\boldsymbol{\lambda}) \boldsymbol{Z}$. We integrate the two kernel matrices $\boldsymbol{K}_{\text{fea}}$ and $\boldsymbol{Z}^\top \boldsymbol{K}_{\text{net}}(\boldsymbol{\lambda}) \boldsymbol{Z}$ by

$$\boldsymbol{K}_{\text{int}}(\boldsymbol{\lambda}) \equiv \boldsymbol{K}_{\text{fea}} \circ \left( \boldsymbol{Z}^\top \boldsymbol{K}_{\text{net}}(\boldsymbol{\lambda}) \boldsymbol{Z} \right), \tag{6}$$

where $\circ$ denotes the *Hadamard product* (a.k.a *element-wise product*). This parameterized matrix $\boldsymbol{K}_{\text{int}}(\boldsymbol{\lambda})$ is guaranteed to be positive semidefinite [6].

Based on the above notations, the dual formulation of Problem 1 can be expressed using the parameterized integrated kernel matrix $\boldsymbol{K}_{\text{int}}(\boldsymbol{\lambda})$ as follows:

**Problem 3.**

$$min \quad \frac{1}{2}\boldsymbol{\alpha}^\top \operatorname{diag}(\boldsymbol{y}) \boldsymbol{K}_{int}(\boldsymbol{\lambda}) \operatorname{diag}(\boldsymbol{y})\boldsymbol{\alpha} - \|\boldsymbol{\alpha}\|_1, \qquad wrt. \quad \boldsymbol{\alpha} \in \mathbb{R}_+^\ell, \text{ and } \boldsymbol{\lambda} \in \mathbb{R}_+^M,$$

$$subj. \ to \quad \boldsymbol{\alpha} \le C_\alpha \boldsymbol{1}_\ell, \quad \boldsymbol{Z} \operatorname{diag}(\boldsymbol{y}) \, \boldsymbol{\alpha} = \boldsymbol{0}_M, \quad \|\boldsymbol{\lambda}\|_1 \le C_\rho. \tag{7}$$

We note that the solutions $\boldsymbol{\alpha}$ and $\boldsymbol{\lambda}$ tend to be sparse due to the $\ell_1$ norm.

Changing the definition of $\boldsymbol{K}_{\text{fea}}$ from the linear kernel to an arbitrary kernel, we can extend the proposed linear MTL method to non-linear domains. Furthermore, we can also deal with non-vectorial structured data by employing a suitable kernel such as the string kernel and the Fisher kernel.

In the test stage, a new sample $\boldsymbol{x}$ in the $j$-th task is classified by

$$f_j(\boldsymbol{x}) = \sum_{t=1}^{\ell} \sum_{i=1}^{M} \alpha_t y_t k_{\text{fea}}(\boldsymbol{x}_t, \boldsymbol{x}) k_{\text{net}}(i,j) Z_{i,t} + b_j, \tag{8}$$

where $k_{\text{fea}}(\cdot, \cdot)$ and $k_{\text{net}}(\cdot, \cdot)$ are the kernel functions over features and tasks, respectively.

## 4.2 Dual MTL Learning by SOCP

Here, we show that the above dual problem can also be reduced to an SOCP. To this end, we first introduce a *matrix-fractional program* (MFP) [7]:

**Problem 4.**

$$min \ (\boldsymbol{F}\boldsymbol{z} + \boldsymbol{g})^\top \boldsymbol{P}(\boldsymbol{z})^{-1} (\boldsymbol{F}\boldsymbol{z} + \boldsymbol{g}) \qquad wrt. \ \boldsymbol{z} \in \mathbb{R}_+^p \quad subj. \ to \ \boldsymbol{P}(\boldsymbol{z}) \equiv \boldsymbol{P}_0 + \sum_{i=1}^{p} z_i \boldsymbol{P}_i \in \mathbb{S}_{++}^n,$$

where $\boldsymbol{P}_i \in \mathbb{S}_+^n$, $\boldsymbol{F} \in \mathbb{R}^{n \times p}$, and $\boldsymbol{g} \in \mathbb{R}^n$. Here $\mathbb{S}_+^n$ and $\mathbb{S}_{++}^n$ denote the positive semidefinite cone and strictly positive definite cone of $n \times n$ matrices, respectively.

Let us re-define $d$ as the rank of the feature kernel matrix $\boldsymbol{K}_{\text{fea}}$. We introduce a matrix $\boldsymbol{V}_{\text{fea}} \in \mathbb{R}^{\ell \times d}$ which decomposes the feature kernel matrix as $\boldsymbol{K}_{\text{fea}} = \boldsymbol{V}_{\text{fea}} \boldsymbol{V}_{\text{fea}}^\top$. Define the $\ell$-dimensional vectors $\boldsymbol{f}_h \in \mathbb{R}^\ell$ of the $h$-th feature as $\boldsymbol{V}_{\text{fea}} \equiv [\boldsymbol{f}_1, \ \ldots, \ \boldsymbol{f}_d] \in \mathbb{R}^{\ell \times d}$ and the matrices $\boldsymbol{F}_h \equiv \boldsymbol{Z} \operatorname{diag}(\boldsymbol{f}_h \circ \boldsymbol{y})$, for $h = 1, \ldots, d$. Using those variables, the objective function in Problem 3 can be rewritten as

$$J_D = \frac{1}{2} \sum_{h=1}^{d} \boldsymbol{\alpha}^\top \boldsymbol{F}_h^\top \left( \frac{1}{M} \boldsymbol{I}_M + \mathcal{U}\boldsymbol{\lambda} \right)^{-1} \boldsymbol{F}_h \boldsymbol{\alpha} - \boldsymbol{\alpha}^\top \boldsymbol{1}_\ell. \tag{9}$$

This implies that Problem 3 can be transformed into the combination of a linear program and $d$ MFPs.

Let us further introduce the vector $\boldsymbol{v}_k \in \mathbb{R}^M$ for each edge: $\boldsymbol{v}_k = \boldsymbol{e}_{i_k} - \boldsymbol{e}_{j_k}$, where $\boldsymbol{e}_{i_k}$ is a unit vector with the $i_k$-th element being one. Let $\boldsymbol{V}_{\mathrm{lap}}$ be a matrix defined by $\boldsymbol{V}_{\mathrm{lap}} = [\boldsymbol{v}_1, \ldots, \boldsymbol{v}_K] \in \mathbb{R}^{M \times K}$. Then we can re-express the graph Lagrangian matrix of tasks as $\mathcal{U}\boldsymbol{\lambda} = \boldsymbol{V}_{\mathrm{lap}} \operatorname{diag}(\boldsymbol{\lambda}) \boldsymbol{V}_{\mathrm{lap}}^\top$.

Given the fact that an MFP can be reduced to an SOCP [7], we can reduce Problem 3 to the following SOCP:

**Problem 5.**

$$min \quad -\mathbf{1}_\ell^\top \boldsymbol{\alpha} + \frac{1}{2} \sum_{h=1}^d s_{0,h} + s_{1,h} + \cdots + s_{K,h}, \tag{10}$$

$$wrt \quad s_{0,h} \in \mathbb{R}, \ s_{k,h} \in \mathbb{R}, \ \boldsymbol{u}_{0,h} \in \mathbb{R}^M, \ \boldsymbol{u}_h = [u_{1,h}, \ldots, u_{K,h}]^\top \in \mathbb{R}^K \quad \forall k, \ \forall h \tag{11}$$

$$\boldsymbol{\lambda} \in \mathbb{R}_+^K, \ \boldsymbol{\alpha} \in \mathbb{R}_+^\ell, \tag{12}$$

$$subj. \ to \quad \boldsymbol{\alpha} \le C_\alpha \mathbf{1}_\ell, \quad \boldsymbol{Z} \operatorname{diag}(\boldsymbol{y}) \boldsymbol{\alpha} = \mathbf{0}_M, \quad \mathbf{1}_K^\top \boldsymbol{\lambda} \le C_\rho, \tag{13}$$

$$M^{-1/2}\boldsymbol{u}_{0,h} + \boldsymbol{V}_{lap}\boldsymbol{u}_h = \boldsymbol{F}_h \boldsymbol{\alpha}, \quad \left\| \begin{bmatrix} 2\boldsymbol{u}_{0,h} \\ s_{0,h} - 1 \end{bmatrix} \right\| \le s_{0,h} + 1, \quad \forall h \tag{14}$$

$$\left\| \begin{bmatrix} 2u_{k,h} \\ s_{k,h} - \lambda_k \end{bmatrix} \right\| \le s_{k,h} + \lambda_k \quad \forall k, \ \forall h \tag{15}$$

Consequently, we obtain the following result:

**Theorem 2.** *The dual problem of CoNs learning (Problem 3) can be reduced to the SOCP in Problem 5 and it can be solved with $O((Kd + \ell)^2((M + K)d + \ell))$ computation.*

## 5 Discussion

In this section, we discuss the properties of the proposed MTL method and the relation to existing methods.

**MTL with Common Bias** A possible variant of the proposed MTL method would be to share the common bias parameter with all tasks (i.e. $b_1 = b_2 = \cdots = b_M$). The idea is expected to be useful particularly when the number of samples in each task is very small. We can also apply the common bias idea in the kernelized version just by replacing the constraint $\boldsymbol{Z} \operatorname{diag}(\boldsymbol{y})\boldsymbol{\alpha} = \mathbf{0}_M$ in Problem 3 by $\boldsymbol{y}^\top \boldsymbol{\alpha} = 0$.

**Global vs. Local Constraints** Micchelli and Pontil [8] have proposed a related MTL method which upper-bounds the *sum* of the differences of $K$ related task pairs, i.e., $\frac{1}{2} \sum_{k=1}^K \|\boldsymbol{w}_{i_k} - \boldsymbol{w}_{j_k}\|^2 \le \rho$. We call it the *global constraint*. This global constraint can also have a similar effect to our local constraint (2), i.e., the related task pairs tend to have close solutions. However, the global constraint can allow some of the distances to be large since only the sum is upper-bounded. This actually causes a significant performance degradation in practice, which will be experimentally demonstrated in Section 6. We note that the idea of local constraints is also used in the kernel learning problem [10].

**Relation to Standard SVMs** By construction, the proposed MTL method includes the standard SVM learning algorithm a special case. Indeed, when the number of tasks is one, Problem 3 is reduced to the standard SVM optimization problem. Thus, the proposed method may be regarded as a natural extension of SVMs.

**Ordinal Regression** As we mentioned in Section 1, MTL approaches are useful in ordinal regression problems. Ordinal regression is a task of learning multiple quantiles, which can be formulated as a set of one-versus-one classification problems. A naive approach to ordinal regression is to individually train $M$ SVMs with score functions $f_i(\boldsymbol{x}) = \langle \boldsymbol{w}_i, \boldsymbol{x} \rangle + b_i$, $i = 1, \ldots, M$. Shashua

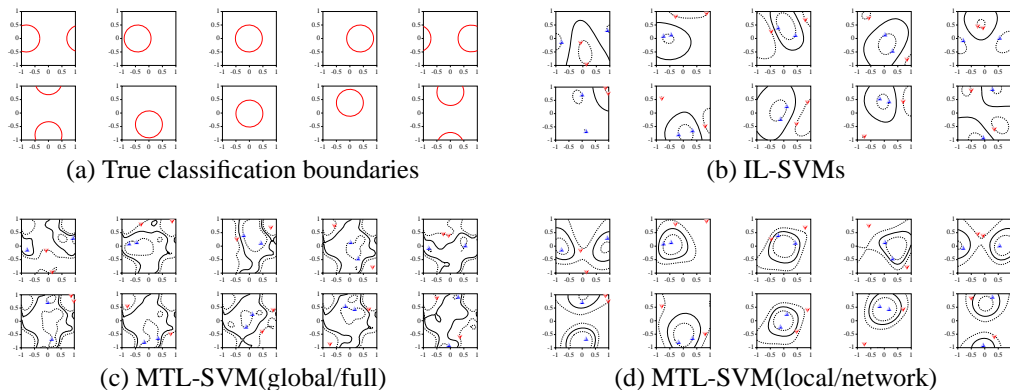

(a) True classification boundaries    (b) IL-SVMs

(c) MTL-SVM(global/full)    (d) MTL-SVM(local/network)

Figure 1: Toy multi classification tasks. Each subfigure contains the 10-th, 30-th, 50-th, 70-th, and 90-th tasks in the top row and the 110-th, 130-th, 150-th, 170-th, and 190-th tasks in the bottom row.

and Levin [9] proposed an ordinal regression method called the *support vector ordinal regression* (SVOR), where the weight vectors are shared by all SVMs (i.e. $\boldsymbol{w}_1 = \boldsymbol{w}_2 = \cdots = \boldsymbol{w}_M$) and only the bias parameter is learned individually.

The proposed MTL method can be naturally employed in ordinal regression by constraining the weight vectors as $\|\boldsymbol{w}_i - \boldsymbol{w}_{i+1}\|^2 \leq \rho$, $i = 1, \ldots, M-1$, i.e., the task network only has a weight between consecutive tasks. This method actually includes the above two ordinal regression approaches as special cases—$C_\rho = 0$ (i.e., ignoring the task network) yields the independent training of SVMs and $C_\rho = \infty$ (i.e., the weight vectors of all SVMs agree) is reduced to SVOR. Thus, in the context of ordinal regression, the proposed method smoothly bridges two extremes and allows us to control the belief of task constraints.

## 6 Experiments

In this section, we show the usefulness of the proposed method through experiments.

### 6.1 Toy Multiple Classification Tasks

First, we illustrate how the proposed method behaves using a 2-dimensional toy data set, which includes 200 tasks (see Figure 1(a)). Each task possesses a circular-shaped classification boundary with different centers and a fixed radius 0.5. The location of the center in the $i$-th task is $(-1 + 0.02(i-1), 0)$ for $1 \leq i \leq 100$ and $(0, -1 + 0.02(i-101))$ for $101 \leq i \leq 200$. For each task, only two positive and two negative samples are generated following the uniform distribution. We construct a task network where consecutive tasks are connected in a circular manner, i.e., $(1, 2)$, $(2, 3)$, ..., $(99, 100)$, and $(100, 1)$ for the first 100 tasks and $(101, 102)$, $(102, 103)$, ..., $(199, 200)$, and $(200, 1)$ for the last 100 tasks; we further add $(50, 150)$, which connects the clusters of the first 100 and the last 100 nodes.

We compare the following methods: a naive method where 200 SVMs are trained indivisually (individually learned SVM, '*IL-SVM*'), the MTL-SVM algorithm where the global constraint and the fully connected task network are used [5] ('*MTL-SVM(global/full)*'), and the proposed method which uses local constraints and the properly defined task network ('*MTL-SVM(local/network)*').

The results are exhibited in Figure 1, showing that IL-SVM can not capture the circular shape due to the small sample size in each task. MTL-SVM(global/full) can successfully capture closed-loop boundaries by making use of the information from other tasks. However, the result is still not so reliable since non-consecutive unrelated tasks heavily damage the solutions. On the other hand, MTL-SVM(local/network) nicely captures the circular boundaries and the results are highly reliable. Thus, given an appropriate task network, the proposed MTL-SVM(local/network) can effectively exploit information of the related tasks.

Table 1: The accuracy of each method in the protein super-family classification task.

| Dataset | IL-SVM | One-SVM | MTL-SVM (global/full) | MTL-SVM (global/network) | MTL-SVM (local/network) |
|---------|--------|---------|-----------------------|--------------------------|-------------------------|
| d-f | 0.908 (0.023) | 0.941 (0.015) | 0.945 (0.013) | 0.933 (0.017) | **0.952** (0.015) |
| d-s | 0.638 (0.067) | 0.722 (0.030) | 0.698 (0.036) | 0.695 (0.032) | **0.747** (0.020) |
| d-o | 0.725 (0.032) | 0.747 (0.017) | 0.748 (0.021) | 0.749 (0.023) | **0.764** (0.028) |
| f-s | 0.891 (0.036) | 0.886 (0.021) | **0.918** (0.020) | 0.911 (0.022) | **0.918** (0.025) |
| f-o | 0.792 (0.046) | 0.819 (0.029) | 0.834 (0.021) | 0.828 (0.015) | **0.838** (0.018) |
| s-o | 0.663 (0.034) | 0.695 (0.034) | 0.692 (0.050) | 0.663 (0.068) | **0.703** (0.036) |

## 6.2 Protein Super-Family Classification

Next, we test the performance of the proposed method with real-world protein super-family classification problems.

The input data are amino acid sequences from the *SCOP database* [1] (not SOCP). We counted 2-mers for extraction of feature vectors. There are 20 kinds of amino acids. Hence, the number of features is $20^2 = 400$. We use RBF kernels, where the kernel width $\sigma^2_{\text{rbf}}$ is set to the average of the squared distances to the fifth nearest neighbors. Each data set consists of two folds. Each fold is divided into several super-families. We here consider the classification problem into the super-families. A positive class is chosen from one fold, and a negative class is chosen from the other fold. We perform multi-task learning from all the possible combinations. For example, three super-families are in DNA/RNA binding, and two in SH3. The number of combinations is $3 \cdot 2 = 6$. So the data set d-s has the six binary classification tasks. We used four folds: DNA/RNA binding, Flavodoxin, OB-fold and SH3. From these folds, we generate six data sets: d-f, d-f, d-o, f-o, f-s, and o-s, where the fold names are abbreviated to d, f, o, and s, respectively.

The task networks are constructed as follows: if the positive super-family or the negative super-family is common to two tasks, the two tasks are regarded as a related task pair and connected by an edge. We compare the proposed MTL-SVM(local/network) with IL-SVM, '*One-SVM*', MTL-SVM(global/full), and MTL-SVM(global/network). One-SVM regards the multiple tasks as one big task and learns the big task once by a standard SVM. We set $C_\alpha = 1$ for all the approaches. The value of the parameter $C_\rho$ for three MTL-SVM approaches is determined by cross-validation over the training set. We randomly pick ten training sequences from each super-family, and use them for training. We compute the classification accuracies of the remaining test sequences. We repeat this procedure 10 times and take the average of the accuracies.

The results are described in Table 1, showing that the proposed MTL-SVM(local/network) compares favorably with the other methods. In this simulation, the task network is constructed rather heuristically. Even so, the proposed MTL-SVM(local/network) is shown to significantly outperform MTL-SVM(global/full), which does not use the network structure. This implies that the proposed method still works well even when the task network contains small errors. It is interesting to note that MTL-SVM(global/network) actually does not work well in this simulation, implying that the task relatedness are not properly controlled by the global constraint. Thus the use of the local constraints would be effective in MTL scenarios.

## 6.3 Ordinal Regression

As discussed in Section 5, MTL methods are useful in ordinal regression. Here we create five ordinal regression data sets described in Table 2, where all the data sets are originally regression and the output values are divided into five quantiles. Therefore, the overall task can be divided into four isolated classification tasks, each of which estimates a quantile. We compare MTL-SVM(local/network) with IL-SVM, *SVOR* [9] (see Section 5), MTL-SVM(full/network) and MTL-SVM(global/network). The value of the parameter $C_\rho$ for three MTL-SVM approaches is determined by cross-validation over the training set. We set $C_\alpha = 1$ for all the approaches. We use RBF kernels, where the parameter $\sigma^2_{\text{rbf}}$ is set to the average of the squared distances to the fifth nearest neighbors. We randomly picked 200 samples for training. The remaining samples are used for evaluating the classification accuracies.

Table 2: The accuracy of each method in ordinal regression tasks.

| Data set | IL-SVM | SVOR | MTL-SVM (global/full) | MTL-SVM (global/network) | MTL-SVM (local/network) |
|---|---|---|---|---|---|
| pumadyn | 0.643 (0.007) | **0.661** (0.006) | 0.629 (0.025) | 0.645 (0.018) | **0.661** (0.007) |
| stock | 0.894 (0.012) | 0.878 (0.011) | 0.872 (0.010) | 0.888 (0.010) | **0.902** (0.007) |
| bank-8fh | **0.781** (0.003) | 0.777 (0.006) | 0.772 (0.006) | 0.773 (0.006) | 0.779 (0.002) |
| bank-8fm | **0.854** (0.004) | 0.845 (0.010) | 0.832 (0.013) | 0.847 (0.009) | **0.854** (0.009) |
| calihouse | 0.648 (0.003) | 0.642 (0.008) | 0.640 (0.005) | 0.646 (0.007) | **0.650** (0.004) |

The averaged performance over five runs is described in Table 2, showing that the proposed MTL-SVM(local/network) is also promising in ordinal regression scenarios.

# 7  Conclusions

In this paper, we proposed a new multi-task learning method, which overcomes the limitation of existing approaches by making use of a task network and local constraints. We demonstrated through simulations that the proposed method is useful in multi-task learning scenario; moreover, it also works excellently in ordinal regression scenarios.

The standard SVMs have a variety of extensions and have been combined with various techniques, e.g., one-class SVMs, SV regression, and the $\nu$-trick. We expect that such extensions and techniques can also be applied similarly to the proposed method. Other possible future works include the elucidation of the entire regularization path and the application to learning from multiple networks; developing algorithms for learning probabilistic models with a task network is also a promising direction to be explored.

**Acknowledgments**

This work was partially supported by a Grant-in-Aid for Young Scientists (B), number 18700287, from the Ministry of Education, Culture, Sports, Science and Technology, Japan.

## Footnotes

[1]More generally, the tasks can be related in an inhomogeneous way, i.e., the *strength* of the relationship among tasks can be dependent on tasks. This general setting can be similarly formulated by a *weighted* network, where edges are weighted according to the strength of the connections. All the discussions in this paper can be easily extended to weighted networks, but for simplicity we focus on unweighted networks.

[2]More generally, an SOCP can include linear equality constraints, but they can be eliminated, for example, by some projection method.

# References

[1] A. Andreeva, D. Howorth, S. E. Brenner, T. J. P. Hubbard, C. Chothia, and A. G. Murzin. SCOP database in 2004: refinements integrate structure and sequence family data. *Nucl. Acid Res.*, 32:D226–D229, 2004.

[2] B. Borchers. CSDP, a C library for semidefinite programming. *Optimization Methods and Software*, 11(1):613–623, 1999.

[3] Stephen Boyd and Lieven Vandenberghe. *Convex Optimization*. Cambridge University Press, 2004.

[4] R. Caruana. Multitask learning. *Machine Learning*, 28(1):41–75, 1997.

[5] T. Evgeniou and M. Pontil. Regularized multitask learning. In *Proc. of 17-th SIGKDD Conf. on Knowledge Discovery and Data Mining*, 2004.

[6] D. Haussler. Convolution kernels on discrete structures. Technical Report UCSC-CRL-99-10, UC Santa Cruz, July 1999.

[7] M. Lobo, L. Vandenberghe, S. Boyd, and H. Lebret. Applications of second-order cone programming. *Linear Algebra and its Applications*, 284:193–228, 1998.

[8] C. A. Micchelli and M. Pontil. Kernels for multi-task learning. In Lawrence K. Saul, Yair Weiss, and Léon Bottou, editors, *Advances in Neural Information Processing Systems 17*, pages 921–928, Cambridge, MA, 2005. MIT Press.

[9] A. Shashua and A. Levin. Ranking with large margin principle: two approaches. In *Advances in Neural Information Processing Systems 15*, pages 937–944, Cambridge, MA, 2003. MIT Press.

[10] K. Tsuda and W.S. Noble. Learning kernels from biological networks by maximizing entropy. *Bioinformatics*, 20(Suppl. 1):i326–i333, 2004.

[11] X. Zhu, J. Kandola, Z. Ghahramani, and J. Lafferty. Nonparametric transforms of graph kernels for semi-supervised learning. In Lawrence K. Saul, Yair Weiss, and Lon Bottou, editors, *Advances in Neural Information Processing Systems 17*, Cambridge, MA, 2004. MIT Press.

